# A Second-Order Translation, Rotation and Scale Invariant Neural Network

Shelly D.D. Goggin          Kristina M. Johnson          Karl E. Gustafson*
Optoelectronic Computing Systems Center and
Department of Electrical and Computer Engineering
University of Colorado at Boulder
Boulder, CO 80309
shellg@boulder.colorado.edu

## ABSTRACT

A second-order architecture is presented here for translation, rotation and scale invariant processing of 2-D images mapped to $n$ input units. This new architecture has a complexity of $O(n)$ weights as opposed to the $O(n^3)$ weights usually required for a third-order, rotation invariant architecture. The reduction in complexity is due to the use of discrete frequency information. Simulations show favorable comparisons to other neural network architectures.

## 1    INTRODUCTION

Multiplicative interactions in neural networks have been proposed (Pitts and Mc-Culloch, 1947; Giles and Maxwell, 1987; McClelland et al, 1988) both to explain biological neural functions and to provide invariances in pattern recognition. Higher-order neural networks are useful for invariant pattern recognition problems, but their complexity prohibits their use in many large image processing applications. The complexity of the third-order rotation invariant neural network of Reid et al, 1990 is $O(n^3)$, which will clearly not scale. For example, when $n$ is on the order of $10^6$, as in high definition television (HDTV), $O(10^{18})$ weights would be required in a third-order neural network. Clearly, image processing applications are best approached with neural networks of lower complexity. We present a translation,

rotation and scale invariant architecture, which has weight complexity of $O(n)$, and requires only multiplicative and additive operations in the activation function.

## 2   HIGHER-ORDER NEURAL NETWORKS

Higher-order neural networks (HONN) have multiplicative terms in their activation function, such that the output of a unit, $o_k$, has the form

$$o_k = f[\sum_{(i=0)}^{(n-1)} \sum_{(j=0)}^{(n-1)} ... \sum_{l=0}^{(n-1)} w_{ij...lk} x_i x_j ... x_l] \qquad (1)$$

where $f$ is a thresholding function, $w_{ij...lk}$ is the weight for each term, and $x_i$ is one of $n$ input values. Some of the $x_i$ could be bias units to give lower order terms. The order of the multiplications is $O(n^m)$ for an m-order network, but the order of the number of weights can be lower. Since the multiplications of data can be done in a preprocessing stage, the major factor in the computational burden is the number of weights. The emphasis on the complexity of the weights is especially relevant for optical implementations of higher-order networks (Psaltis et al, 1988, Zhang et al, 1990), since the multiplications can usually be performed in parallel.

Invariances can be achieved with higher-order neural networks by using the spatial frequencies of the input as a priori information. Wechsler and Zimmerman, 1988, compute the Fourier transform of the data in polar coordinates and use these data as inputs to a neural network to achieve rotation, scale and translation invariance. The disadvantage with this approach is that the Fourier transform and the computation of polar coordinates require more complex operations than addition and multiplication of inputs. It has been shown that second-order networks can be constructed to provide either translation and scale invariance or rotation and scale invariance (Giles et al, 1988). However, their approach does not consider the difficulties in defining scale and rotation for images made up of pixels. Our architecture directly addresses the problem of rotation, translation and scale invariance in pattern recognition for 2-D arrays of binary pixels. Restrictions permit structure to be built into the weights, which reduces their complexity.

## 3   WEDGE-RING HONN

We present a new architecture for a second-order neural network based on the concept of the wedge-ring detector (Casasent, 1985). When a wedge-ring detector is used in the Fourier plane of an optical processor, a set of features are obtained that are invariant to scale, rotation and translation. As shown in figure 1, the lens performs a spatial Fourier transform on an image, which yields an intensity pattern that is invariant to translations in the image plane. The ring detectors sum the amplitudes of the spatial frequencies with the same radial distance from the zero frequency, to give features that are invariant to rotation and shift changes. The wedge detectors sum the amplitudes of frequencies within a range of angles with respect to the zero frequency to produce features that are invariant to scale and shift changes, assuming the images retain the same zero frequency power as they are scaled.

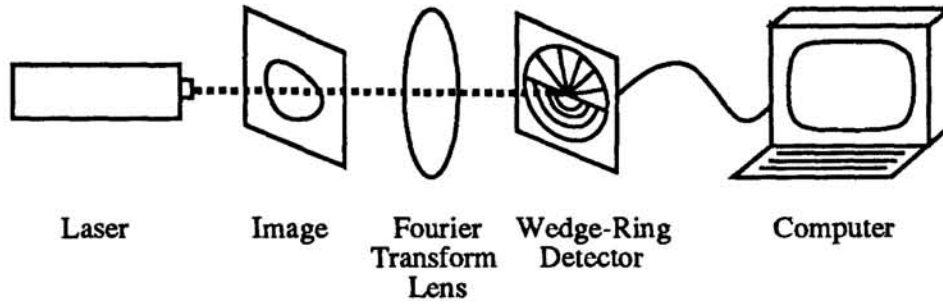

Laser    Image    Fourier    Wedge-Ring    Computer
                  Transform  Detector
                  Lens

Figure 1: A Wedge-Ring Detector Optical Processor

In a multi-pixel, binary image, a second-order neural network can perform the same function as the wedge-ring detector without the need for a Fourier transform. For an image of dimensions $\sqrt{n} \times \sqrt{n}$, let us define the pixel spatial frequency $f_{i,j}$ as

$$f_{k,l} = \sum_{(i=0)}^{(\sqrt{n}-1-|k|)} \sum_{(j=0)}^{(\sqrt{n}-1-|l|)} x_{i,j} x_{i+|k|,j+|l|}, \quad -(\sqrt{n}-1) \leq k, \, l \leq \sqrt{n}-1 \quad (2)$$

where $x_{i,j}$ is a binary valued pixel at location $(i, j)$. Note that the pixel frequencies have symmetry; $f_{i,j} = f_{-i,-j}$. The frequency terms can be arranged in a grid in a manner analogous to the Fourier transform image in the optical wedge-ring detector. (See figure 2.)

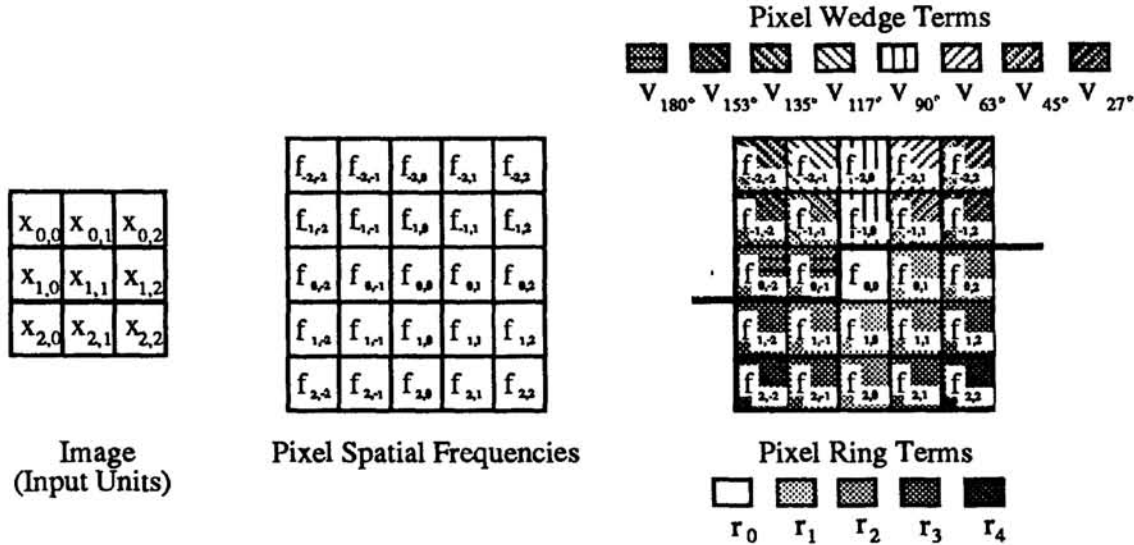

Image
(Input Units)

Pixel Spatial Frequencies

Pixel Wedge Terms

$V_{180°}$ $V_{153°}$ $V_{135°}$ $V_{117°}$ $V_{90°}$ $V_{63°}$ $V_{45°}$ $V_{27°}$

Pixel Ring Terms

$r_0$ $r_1$ $r_2$ $r_3$ $r_4$

Figure 2: A Simple Input Image and its Associated Pixel Spatial Frequencies, Pixel Ring Terms and Pixel Wedge Terms

For all integers $p$, $0 \leq p \leq 2(\sqrt{n}-1)$, the ring pixel terms $r_p$ are given by

$$r_p = 2 \sum_{|k|+|l|=p} f_{k,l}, \quad \begin{array}{l} 0 \leq k \leq \sqrt{n}-1, \; 0 \leq l \leq \sqrt{n}-1, \; if \; k = 0. \\ -(\sqrt{n}-1) \leq l \leq \sqrt{n}-1, \; if \; k > 0. \end{array} \quad (3)$$

as shown in figure 2. This definition of the ring pixel terms works well for images with a small number of pixels. Larger pixel arrays can use the following

definition. For $0 \leq p \leq 2(\sqrt{n}-1)^2$,

$$r_p = 2 \sum_{k^2+l^2=p} f_{k,l}, \quad 0 \leq k \leq \sqrt{n}-1, \quad \begin{array}{l} 0 \leq l \leq \sqrt{n}-1, \; if \; k=0. \\ -(\sqrt{n}-1) \leq l \leq \sqrt{n}-1, \; if \; k>0. \end{array} \tag{4}$$

Note that $p$ will not take on all values less than $2n$. The number of ring pixel terms generated by equation 4 is less than or equal to $\lceil n/2 \rceil + \lfloor \sqrt{n}/2 \rfloor$. The number of ring pixel terms can be reduced by making the rings a fixed width, $\Delta r$. Then, for all integers $p$, $0 \leq p \leq \lceil \sqrt{2}(\sqrt{n}-1)/\Delta r \rceil$.

$$r_p = 2 \sum_{(p-1)\Delta r < \sqrt{k^2+l^2} \leq p\Delta r} f_{k,l}, \quad \begin{array}{l} 0 \leq k \leq \sqrt{n}-1, \\ 0 \leq l \leq \sqrt{n}-1, \; if \; k=0. \\ -(\sqrt{n}-1) \leq l \leq \sqrt{n}-1, \; if \; k>0. \end{array} \tag{5}$$

As the image size increases, the ring pixel terms will approximate continuous rings.

For $0 < \theta \leq 180°$, the wedge pixel terms $v_\theta$ are

$$v_\theta = 2 \sum_{\tan^{-1}(k/l)=\theta} f_{k,l}, \quad -(\sqrt{n}-1) \leq k \leq 0, \quad \begin{array}{l} -(\sqrt{n}-1) \leq l \leq 1, \; if \; k=0, \\ -(\sqrt{n}-1) \leq l \leq \sqrt{n}-1, \; if \; k<0, \end{array} \tag{6}$$

as shown in figure 2. The number of wedge pixel terms is less than or equal to $2n - 2\sqrt{n} + 1$. The number of wedge pixel terms can be reduced by using a fixed wedge width, $\Delta v$. Then for all integers q, $1 \leq q \leq \lceil 180°/\Delta v \rceil$,

$$v_\theta = 2 \sum_{(q-1)\Delta v < \tan^{-1}(k/l) \leq q\Delta v} f_{k,l}, \quad \begin{array}{l} -(\sqrt{n}-1) \leq k \leq 0, \\ -(\sqrt{n}-1) \leq l \leq 1, \; if \; k=0, \\ -(\sqrt{n}-1) \leq l \leq \sqrt{n}-1, \; if \; k<0, \end{array} \tag{7}$$

For small pixel arrays, the pixel frequencies are not evenly distributed between the wedges.

All of the operations from the second-order terms to the pixel frequencies and from the pixel frequencies to the ring and wedge pixel terms are linear. Therefore, the values of the wedge-ring features can be obtained by directly summing the second-order terms, without explicitly determining the individual spatial frequencies.

$$r_p = 2 \sum_{(k^2+l^2=p)} \sum_{(i=0)}^{(\sqrt{n}-1-|k|)} \sum_{(j=0)}^{(\sqrt{n}-1-|l|)} x_{i,j} x_{i+|k|,j+|l|}, \quad \begin{array}{l} 0 \leq k \leq \sqrt{n}-1, \\ 0 \leq l \leq \sqrt{n}-1, \; if \; k=0. \\ -(\sqrt{n}-1) \leq l \leq \sqrt{n}-1, \\ if \; k>0. \end{array} \tag{8}$$

$$v_\theta = 2 \sum_{(\tan^{-1}(k/l)=\theta)} \sum_{(i=0)}^{(\sqrt{n}-1-|k|)} \sum_{(j=0)}^{(\sqrt{n}-1-|l|)} x_{i+|k|,j+|l|} x_{i,j}, \quad \begin{array}{l} -(\sqrt{n}-1) \leq k \leq 0, \\ -(\sqrt{n}-1) \leq l \leq 1, \\ if \; k=0. \\ -(\sqrt{n}-1) \leq l \leq \sqrt{n}-1, \\ if \; k<0. \end{array} \tag{9}$$

A mask can be used to sum the second-order terms directly. For an example of the mask for the $3 \times 3$ image, see figure 3.

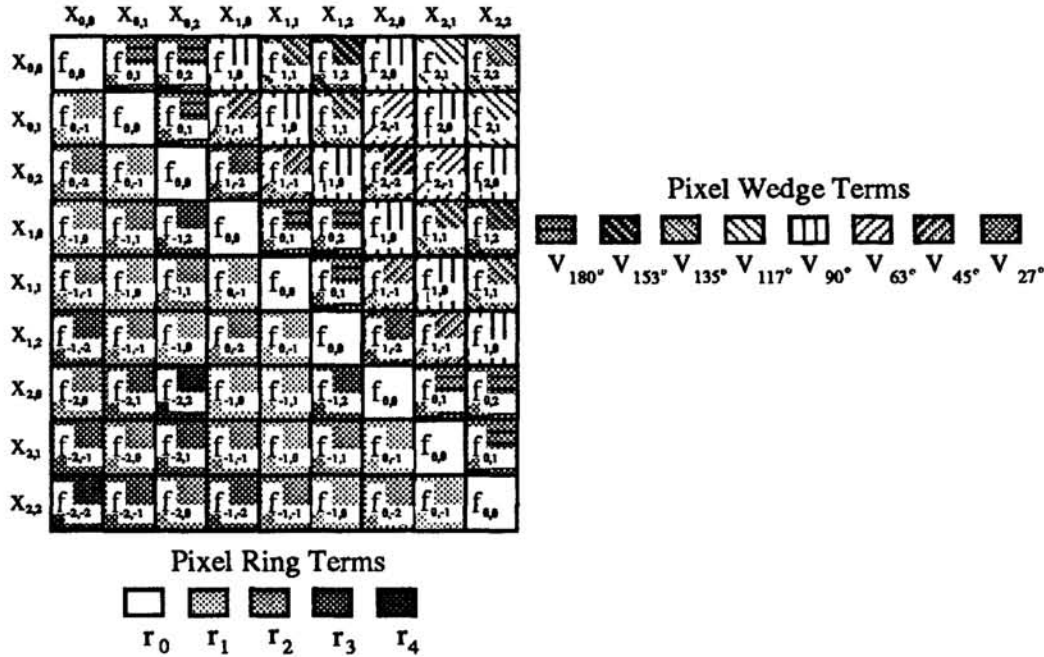

Pixel Wedge Terms

$V_{180°}$ $V_{153°}$ $V_{135°}$ $V_{117°}$ $V_{90°}$ $V_{63°}$ $V_{45°}$ $V_{27°}$

Pixel Ring Terms

$r_0$  $r_1$  $r_2$  $r_3$  $r_4$

Figure 3: A Mask for Summing Second-Order Terms for Ring Features
and Wedge Features for the Image in Figure 2

The ring and wedge pixel terms can be used as inputs for a multilayer neural network that can then perform pattern recognition with general combinations of these features. The output of the first (and possibly only) hidden layer units are for unit $j$,

$$o_j = f[\sum_p w_{j,p} r_p + \sum_\theta w_{j,\theta} v_\theta], \tag{10}$$

where $f$ here is the threshold function. The total number of ring and wedge terms, which corresponds to the number of weights, is less than or equal to $(5/2)n$.

## 4  EXAMPLE RESULTS FOR THE TC PROBLEM

Results have been obtained for the $9 \times 9$ TC problem (McClelland et al, 1988) (see figure 4). Since wedge and ring pixel terms are used, a solution to the problem is readily seen. Figure 5 shows the final neural network architecture. Equations 4 and 6 are used to calculate the ring and wedge pixel terms, respectively. With two additional layers, the network can distinguish between the T and the C at any of the three scales or four rotations. In the hidden layer, the $180°$ wedge pixel term is subtracted from the $90°$ wedge pixel term and vice-versa with a bias unit weighted by 0.5 and a hard-limiting threshold function. This computation results in hidden units with values $(0, 1)$ or $(1, 0)$ for the C and $(1, 1)$ for the T. The next level then performs a binary AND, to get a 1 for T and a 0 for C. The wedge features are also used in a layer to determine whether the image was rotated by $\pm 90°$ or not. The ring units are used as input to a layer with an output unit for each of the three scales. Due to the reduced complexity of the weights in this second-order neural network, a solution for the architecture and weights is obtained by inspection, whereas the

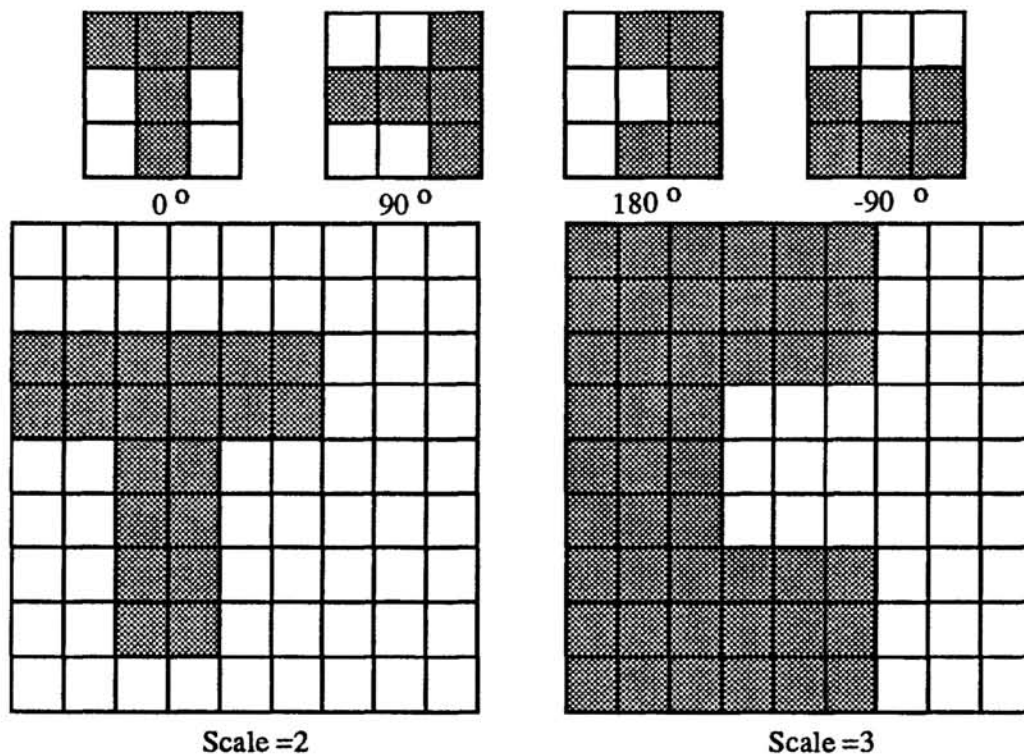

Figure 4: Examples of Rotated and Scaled Input Images for the TC Problem

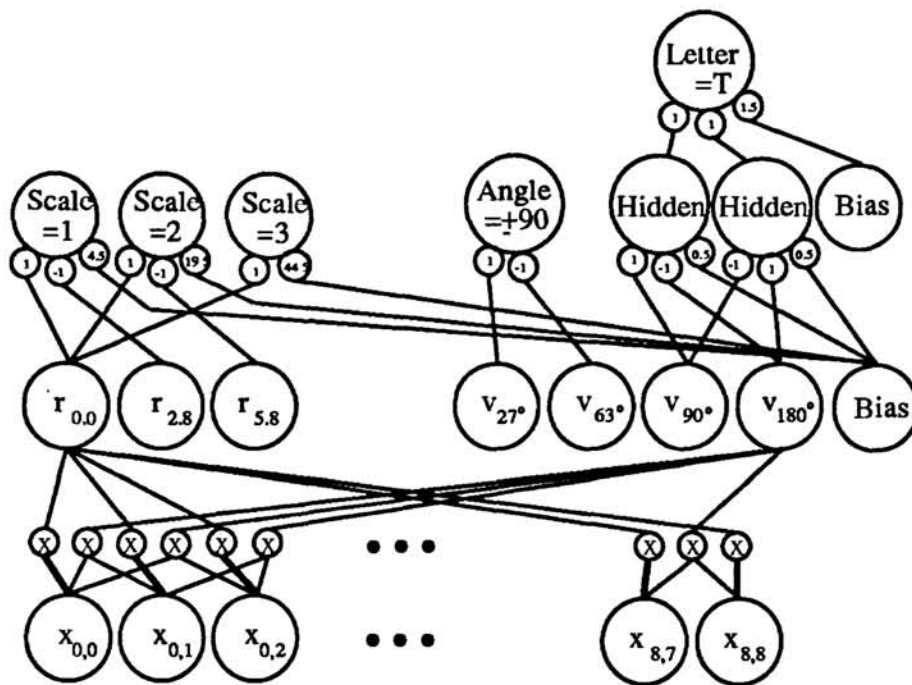

Figure 5: Multilayer Neural Network for the Wedge-Ring Features for the TC Problem

same problem required computer simulation when presented to a third-order neural network (Reid et al, 1990).

## 5  CONCLUSIONS

In this paper, we show how the weight complexity in a higher-order neural network is reduced from $O(n^3)$ to $O(n)$ by building into the architecture invariances in rotation, translation and scale. These invariances were built into the neural network architecture by analogy to the architecture for feature extraction in the optical wedge-ring detector system. This neural network architecture has been shown to greatly simplify the computations required to solve the classic TC problem.

### Acknowledgements

We gratefully acknowledge fellowship support from GTE Research Labs and the NSF Engineering Research Center for Optoelectronic Computing Systems grant CDR8622236.

## Footnotes

*Department of Mathematics

### References

D. Casasent, "Coherent optical pattern recognition: A review," *Optical Engineering,* vol. 24, no. 1, pp. 26-32 (1985).

C.L. Giles, R.D. Griffin, and T. Maxwell, "Encoding geometric invariances in higher-order networks," *In: Neural Information Processing Systems,* D. Z. Anderson (ed.), (New York: American Institute of Physics, 1988) pp. 301-309.

C.L. Giles and T. Maxwell, "Learning, invariance and generalization in high-order neural networks," *Applied Optics,* vol. 26, no. 23, pp. 4972-4978 (1987).

J.L. McClelland, D.E. Rumelhart and the PDP Research Group, *Parallel Distributed Processing, Explorations in the Microstructure of Cognition,* (Cambridge, MA: The MIT Press, 1988).

W. Pitts and W.S. McCulloch, "How we know universals: The perception of auditory and visual forms," *Bulletin of Mathematical Biophysics,* vol. 9, pp. 127-147 (1947).

D. Psaltis, C.H. Park and J. Hong, "Higher order associative memories and their optical implementations," *Neural Networks,* vol. 1, pp. 149-163 (1988).

M.B. Reid, L. Spirkovska and E. Ochoa, "Simultaneous position, scale and rotation invariant pattern classification using third-order neural networks," *To appear in: The International Journal of Neural Networks - Research and Applications.*

H. Wechsler and G.L. Zimmerman, "Invariant object recognition using a distributed associative memory," *In: Neural Information Processing Systems,* D. Z. Anderson (ed.), (New York: American Institute of Physics, 1988) pp. 830-839.

L. Zhang, M.G. Robinson and K.M. Johnson, "Optical implementation of a second order neural network," *International Neural Network Conference,* Paris, July, 1990.